# Learning with Hypergraphs: Clustering, Classification, and Embedding

**Dengyong Zhou[†], Jiayuan Huang[‡], and Bernhard Schölkopf[§]**
[†]NEC Laboratories America, Inc.
4 Independence Way, Suite 200, Princeton, NJ 08540, USA
[‡]School of Computer Science, University of Waterloo
Waterloo ON, N2L3G1, Canada
[§]Max Planck Institute for Biological Cybernetics
Spemannstr. 38, 72076 Tübingen, Germany
{dengyong.zhou, jiayuan.huang, bernhard.schoelkopf}@tuebingen.mpg.de

## Abstract

We usually endow the investigated objects with pairwise relationships, which can be illustrated as graphs. In many real-world problems, however, relationships among the objects of our interest are more complex than pairwise. Naively squeezing the complex relationships into pairwise ones will inevitably lead to loss of information which can be expected valuable for our learning tasks however. Therefore we consider using hypergraphs instead to completely represent complex relationships among the objects of our interest, and thus the problem of learning with hypergraphs arises. Our main contribution in this paper is to generalize the powerful methodology of spectral clustering which originally operates on undirected graphs to hypergraphs, and further develop algorithms for hypergraph embedding and transductive classification on the basis of the spectral hypergraph clustering approach. Our experiments on a number of benchmarks showed the advantages of hypergraphs over usual graphs.

## 1 Introduction

In machine learning problem settings, we generally assume pairwise relationships among the objects of our interest. An object set endowed with pairwise relationships can be naturally illustrated as a graph, in which the vertices represent the objects, and any two vertices that have some kind of relationship are joined together by an edge. The graph can be undirected or directed. It depends on whether the pairwise relationships among objects are symmetric or not. A finite set of points in Euclidean space associated with a kernel matrix is a typical example of undirected graphs. As to directed graphs, a well-known instance is the World Wide Web. A hyperlink can be thought of as a directed edge because given an arbitrary hyperlink we cannot expect that there certainly exists an inverse one, that is, the hyperlink based relationships are asymmetric [20].

However, in many real-world problems, representing a set of complex relational objects as undirected or directed graphs is not complete. For illustrating this point of view, let us consider a problem of grouping a collection of articles into different topics. Given an article, assume the only information that we have is who wrote this article. One may construct an undirected graph in which two vertices are joined together by an edge if there is at least one common author of their corresponding articles (Figure 1), and then an undirected graph based clustering approach is applied, e.g. spectral graph techniques [7, 11, 16]. The undirected graph may be further embellished by assigning to each edge a weight equal to the

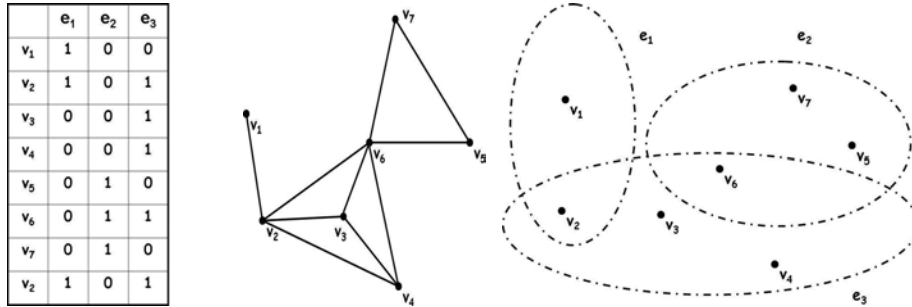

Figure 1: Hypergraph vs. simple graph. Left: an author set $E = \{e_1, e_2, e_3\}$ and an article set $V = \{v_1, v_2, v_3, v_4, v_5, v_6, v_7\}$. The entry $(v_i, e_j)$ is set to 1 if $e_j$ is an author of article $v_i$, and 0 otherwise. Middle: an undirected graph in which two articles are joined together by an edge if there is at least one author in common. This graph cannot tell us whether the same person is the author of three or more articles or not. Right: a hypergraph which completely illustrates the complex relationships among authors and articles.

number of authors in common. The above method may sound natural, but within its graph representation we obviously miss the information on whether the same person joined writing three or more articles or not. Such information loss is unexpected because the articles by the same person likely belong to the same topic and hence the information is useful for our grouping task.

A natural way of remedying the information loss issue occurring in the above methodology is to represent the data as a hypergraph instead. A hypergraph is a graph in which an edge can connect more than two vertices [2]. In other words, an edge is a subset of vertices. In what follows, we shall unifiedly refer to the usual undirected or directed graphs as simple graphs. Moreover, without special mentioning, the referred simple graphs are undirected. It is obvious that a simple graph is a special kind of hypergraph with each edge containing two vertices only. In the problem of clustering articles stated before, it is quite straightforward to construct a hypergraph with the vertices representing the articles, and the edges the authors (Figure 1). Each edge contains all articles by its corresponding author. Even more than that, we can consider putting positive weights on the edges to encode our prior knowledge on authors' work if we have. For instance, for a person working on a broad range of fields, we may assign a relatively small value to his corresponding edge.

Now we can completely represent the complex relationships among objects by using hypergraphs. However, a new problem arises. How to partition a hypergraph? This is the main problem that we want to solve in this paper. A powerful technique for partitioning simple graphs is spectral clustering. Therefore, we generalize spectral clustering techniques to hypergraphs, more specifically, the normalized cut approach of [16]. Moreover, as in the case of simple graphs, a real-valued relaxation of the hypergraph normalized cut criterion leads to the eigendecomposition of a positive semidefinite matrix, which can be regarded as an analogue of the so-called Laplacian for simple graphs (cf. [5]), and hence we suggestively call it the hypergraph Laplacian. Consequently, we develop algorithms for hypergraph embedding and transductive inference based on the hypergraph Laplacian.

There have actually existed a large amount of literature on hypergraph partitioning, which arises from a variety of practical problems, such as partitioning circuit netlists [11], clustering categorial data [9], and image segmentation [1]. Unlike the present work however, they generally transformed hypergraphs to simple ones by using the heuristics we discussed in the beginning or other domain-specific heuristics, and then applied simple graph based spectral clustering techniques. [9] proposed an iterative approach which was indeed designed for hypergraphs. Nevertheless it is not a spectral method. In addition, [6] and [17] considered propagating label distributions on hypergraphs.

The structure of the paper is as follows. We first introduce some basic notions on hypergraphs in Section 2. In Section 3, we generalize the simple graph normalized cut to

hypergraphs. As shown in Section 4, the hypergraph normalized cut has an elegant probabilistic interpretation based on a random walk naturally associated with a hypergraph. In Section 5, we introduce the real-valued relaxation to approximately obtain hypergraph normalized cuts, and also the hypergraph Laplacian derived from this relaxation. In section 6, we develop a spectral hypergraph embedding technique based on the hypergraph Laplacian. In Section 7, we address transductive inference on hypergraphs, this is, classifying the vertices of a hypergraph provided that some of its vertices have been labeled. Experimental results are shown in Section 8, and we conclude this paper in Section 9.

## 2 Preliminaries

Let $V$ denote a finite set of objects, and let $E$ be a family of subsets $e$ of $V$ such that $\cup_{e \in E} = V$. Then we call $G = (V, E)$ a *hypergraph* with the *vertex* set $V$ and the *hyperedge* set $E$. A hyperedge containing just two vertices is a simple graph edge. A *weighted hypergraph* is a hypergraph that has a positive number $w(e)$ associated with each hyperedge $e$, called the *weight* of hyperedge $e$. Denote a weighted hypergraph by $G = (V, E, w)$. A hyperedge $e$ is said to be *incident* with a vertex $v$ when $v \in e$. For a vertex $v \in V$, its *degree* is defined by $d(v) = \sum_{\{e \in E | v \in e\}} w(e)$. Given an arbitrary set $S$, let $|S|$ denote the cardinality of $S$. For a hyperedge $e \in E$, its degree is defined to be $\delta(e) = |e|$. We say that there is a *hyperpath* between vertices $v_1$ and $v_k$ when there is an alternative sequence of distinct vertices and hyperedges $v_1, e_1, v_2, e_2, \ldots, e_{k-1}, v_k$ such that $\{v_i, v_{i+1}\} \subseteq e_i$ for $1 \leq i \leq k-1$. A hypergraph is *connected* if there is a path for every pair of vertices. In what follows, the hypergraphs we mention are always assumed to be connected. A hypergraph $G$ can be represented by a $|V| \times |E|$ matrix $H$ with entries $h(v, e) = 1$ if $v \in e$ and 0 otherwise, called the *incidence matrix* of $G$. Then $d(v) = \sum_{e \in E} w(e) h(v, e)$ and $\delta(e) = \sum_{v \in V} h(v, e)$. Let $D_v$ and $D_e$ denote the diagonal matrices containing the vertex and hyperedge degrees respectively, and let $W$ denote the diagonal matrix containing the weights of hyperedges. Then the *adjacency matrix* $A$ of hypergraph $G$ is defined as $A = HWH^T - D_v$, where $H^T$ is the transpose of $H$.

## 3 Normalized hypergraph cut

For a vertex subset $S \subset V$, let $S^c$ denote the compliment of $S$. A cut of a hypergraph $G = (V, E, w)$ is a partition of $V$ into two parts $S$ and $S^c$. We say that a hyperedge $e$ is cut if it is incident with the vertices in $S$ and $S^c$ simultaneously.

Given a vertex subset $S \subset V$, define the *hyperedge boundary* $\partial S$ of $S$ to be a hyperedge set which consists of hyperedges which are cut, i.e. $\partial S := \{e \in E | e \cap S \neq \emptyset, e \cap S^c \neq \emptyset\}$, and define the *volume* $\mathrm{vol}\, S$ of $S$ to be the sum of the degrees of the vertices in $S$, that is, $\mathrm{vol}\, S := \sum_{v \in S} d(v)$. Moreover, define the volume of $\partial S$ by

$$\mathrm{vol}\, \partial S := \sum_{e \in \partial S} w(e) \frac{|e \cap S|\, |e \cap S^c|}{\delta(e)}. \tag{1}$$

Clearly, we have $\mathrm{vol}\, \partial S = \mathrm{vol}\, \partial S^c$. The definition given by Equation (1) can be understood as follows. Let us imagine each hyperedge $e$ as a clique, i.e. a fully connected subgraph. For avoiding unnecessary confusion, we call the edges in such an imaginary subgraph the subedges. Moreover, we assign the same weight $w(e)/\delta(e)$ to all subedges. Then, when a hyperedge $e$ is cut, there are $|e \cap S|\, |e \cap S^c|$ subedges are cut, and hence a single sum term in Equation (1) is the sum of the weights over the subedges which are cut. Naturally, we try to obtain a partition in which the connection among the vertices in the same cluster is dense while the connection between two clusters is sparse. Using the above introduced definitions, we may formalize this natural partition as

$$\underset{\emptyset \neq S \subset V}{\operatorname{argmin}} c(S) := \mathrm{vol}\, \partial S \left( \frac{1}{\mathrm{vol}\, S} + \frac{1}{\mathrm{vol}\, S^c} \right). \tag{2}$$

For a simple graph, $|e \cap S| = |e \cap S^c| = 1$, and $\delta(e) = 2$. Thus the right-hand side of Equation (2) reduces to the simple graph normalized cut [16] up to a factor $1/2$. In what follows, we explain the hypergraph normalized cut in terms of random walks.

## 4  Random walk explanation

We associate each hypergraph with a natural random walk which has the transition rule as follows. Given the current position $u \in V$, first choose a hyperedge $e$ over all hyperedges incident with $u$ with the probability proportional to $w(e)$, and then choose a vertex $v \in e$ uniformly at random. Obviously, it generalizes the natural random walk defined on simple graphs. Let $P$ denote the transition probability matrix of this hypergraph random walk. Then each entry of $P$ is

$$p(u,v) = \sum_{e \in E} w(e) \frac{h(u,e)}{d(u)} \frac{h(v,e)}{\delta(e)}. \tag{3}$$

In matrix notation, $P = D_v^{-1} H W D_e^{-1} H^T$. The stationary distribution $\pi$ of the random walk is

$$\pi(v) = \frac{d(v)}{\operatorname{vol} V}, \tag{4}$$

which follows from that

$$
\begin{aligned}
\sum_{u \in V} \pi(u) p(u,v) &= \sum_{u \in V} \frac{d(u)}{\operatorname{vol} V} \sum_{e \in E} \frac{w(e) h(u,e) h(v,e)}{d(u) \delta(e)} = \frac{1}{\operatorname{vol} V} \sum_{u \in V} \sum_{e \in E} \frac{w(e) h(u,e) h(v,e)}{\delta(e)} \\
&= \frac{1}{\operatorname{vol} V} \sum_{e \in E} w(e) \sum_{u \in V} h(u,e) \frac{h(v,e)}{\delta(e)} = \frac{1}{\operatorname{vol} V} \sum_{e \in E} w(e) h(v,e) = \frac{d(v)}{\operatorname{vol} V}.
\end{aligned}
$$

We written $c(S) = \dfrac{\operatorname{vol} \partial S}{\operatorname{vol} V} \left( \dfrac{1}{\operatorname{vol} S / \operatorname{vol} V} + \dfrac{1}{\operatorname{vol} S^c / \operatorname{vol} V} \right)$. From Equation (4), we have

$$\frac{\operatorname{vol} S}{\operatorname{vol} V} = \sum_{v \in S} \frac{d(v)}{\operatorname{vol} V} = \sum_{v \in V} \pi(v), \tag{5}$$

that is, the ratio $\operatorname{vol} S / \operatorname{vol} V$ is the probability with which the random walk occupies some vertex in $S$. Moreover, from Equations (3) and (4), we have

$$
\begin{aligned}
\frac{\operatorname{vol} \partial S}{\operatorname{vol} V} &= \sum_{e \in \partial S} \frac{w(e)}{\operatorname{vol} V} \frac{|e \cap S| |e \cap S^c|}{\delta(e)} = \sum_{e \in \partial S} \sum_{u \in e \cap S} \sum_{v \in e \cap S^c} \frac{w(e)}{\operatorname{vol} V} \frac{h(u,e) h(v,e)}{\delta(e)} \\
&= \sum_{e \in \partial S} \sum_{u \in e \cap S} \sum_{v \in e \cap S^c} w(e) \frac{d(u)}{\operatorname{vol} V} \frac{h(u,e)}{d(u)} \frac{h(v,e)}{\delta(e)} \\
&= \sum_{u \in S} \sum_{v \in S^c} \frac{d(u)}{\operatorname{vol} V} \sum_{e \in S} w(e) \frac{h(u,e)}{d(u)} \frac{h(v,e)}{\delta(e)} = \sum_{u \in S} \sum_{v \in S^c} \pi(u) p(u,v),
\end{aligned}
$$

that is, the ratio $\operatorname{vol} \partial S / \operatorname{vol} V$ is the probability with which one sees a jump of the random walk from $S$ to $S^c$ under the stationary distribution. From Equations (5) and (6), we can understand the hypergraph normalized cut criterion as follows: looking for a cut such that the probability with which the random walk crosses different clusters is as small as possible while the probability with which the random walk stays in the same cluster is as large as possible. It is worth pointing out that the random walk view is consistent with that for the simple graph normalized cut [13]. The consistency means that our generalization of the normalized cut approach from simple graphs to hypergraphs is reasonable.

## 5  Spectral hypergraph partitioning

As in [16], the combinatorial optimization problem given by Equation (2) is NP-complete, and it can be relaxed (2) into a real-valued optimization problem

$$\operatorname*{argmin}_{f \in \mathbb{R}^{|V|}} \frac{1}{2} \sum_{e \in E} \sum_{\{u,v\} \subseteq e} \frac{w(e)}{\delta(e)} \left( \frac{f(u)}{\sqrt{d(u)}} - \frac{f(v)}{\sqrt{d(v)}} \right)^2$$

$$\text{subject to } \sum_{v \in V} f^2(v) = 1, \ \sum_{v \in V} f(v) \sqrt{d(v)} = 0.$$

We define the matrices $\Theta = D_v^{-1/2} H W D_e^{-1} H^T D_v^{-1/2}$ and $\Delta = I - \Theta$, where $I$ denotes the identity matrix. Then it can be verified that

$$\sum_{e \in E} \sum_{\{u,v\} \subseteq e} \frac{w(e)}{\delta(e)} \left( \frac{f(u)}{\sqrt{d(u)}} - \frac{f(v)}{\sqrt{d(v)}} \right)^2 = 2 f^T \Delta f.$$

Note that this also shows that $\Delta$ is positive semi-definite. We can check that the smallest eigenvalue of $\Delta$ is 0, and its corresponding eigenvector is just $\sqrt{d}$. Therefore, from standard results in linear algebra, we know that the solution to the optimization problem is an eigenvector $\Phi$ of $\Delta$ associated with its smallest nonzero eigenvalue. Hence, the vertex set is clustered into the two parts $S = \{v \in V | \Phi(v) \geq 0\}$ and $S^c = \{v \in V | \Phi(v) < 0\}$. For a simple graph, the edge degree matrix $D_e$ reduces to $2I$. Thus

$$\Delta = I - \frac{1}{2} D_v^{-1/2} H W H^T D_v^{-1/2} = I - \frac{1}{2} D_v^{-1/2} \left( D_v + A \right) D_v^{-1/2} = \frac{1}{2} \left( I - D_v^{-1/2} A D_v^{-1/2} \right),$$

which coincides with the simple graph Laplacian up to a factor of $1/2$. So we suggestively call $\Delta$ the hypergraph Laplacian.

As in [20] where the spectral clustering methodology is generalized from undirected to directed simple graphs, we may consider generalizing the present approach to *directed hypergraphs* [8]. A directed hypergraph is a hypergraph in which each hyperedge $e$ is an ordered pair $(X, Y)$ where $X \subseteq V$ is the *tail* of $e$ and $Y \subseteq V \setminus X$ is the *head*. Directed hypergraphs been used to model various practical problems from biochemical networks [15] to natural language parsing [12].

## 6 Spectral hypergraph embedding

As in the simple graph case [4, 10], it is straightforward to extend the spectral hypergraph clustering approach to $k$-way partitioning. Denote a $k$-way partition by $(V_1, \cdots, V_k)$, where $V_1 \cup V_2 \cup \cdots \cup V_k = V$, and $V_i \cap V_j = \emptyset$ for all $1 \leq i, j \leq k$. We may obtain a $k$-way partition by minimizing $c(V_1, \cdots, V_k) = \sum_{i=1}^{k} \frac{\text{vol} \, \partial V_i}{\text{vol} \, V_i}$ over all $k$-way partitions. Similarly, the combinatorial optimization problem can be relaxed into a real-valued one, of which the solution can be any orthogonal basis of the linear space spanned by the eigenvectors of $\Delta$ associated with the $k$ smallest eigenvalues.

**Theorem 1.** *Assume a hypergraph $G = (V, E, w)$ with $|V| = n$. Denote the eigenvalues of the Laplacian $\Delta$ of $G$ by $\lambda_1 \leq \lambda_2 \leq \cdots \leq \lambda_n$. Define $c_k(G) = \min c(V_1, \cdots, V_k)$, where the minimization is over all $k$-way partitions. Then $\sum_{i=1}^{k} \lambda_i \leq c_k(G)$.*

*Proof.* Let $r_i$ be a $n$-dimensional vector defined by $r_i(v) = 1$ if $v \in V_i$, and 0 otherwise. Then

$$c(V_1, \cdots, V_k) = \sum_{i=1}^{k} \frac{r_i^T (D_v - H W D_e^{-1} H^T) r_i}{r_i^T D_v r_i}$$

Define $s_i = D_v^{-1/2} r_i$, and $f_i = s_i / \|s_i\|$, where $\| \cdot \|$ denotes the usual Euclidean norm. Thus

$$c(V_1, \cdots, V_k) = \sum_{i=1}^{k} f_i^T \Delta f_i = \text{tr} \, F^T \Delta F,$$

where $F = [f_1 \cdots f_k]$. Clearly, $F^T F = I$. If allowing the elements of $r_i$ to take arbitrary continuous values rather than Boolean ones only, we have

$$c_k(G) = \min c(V_1, \cdots, V_k) \geq \min_{F^T F = I} \text{tr} \, F^T \Delta F = \sum_{i=1}^{k} \lambda_i.$$

The last equality is from standard results in linear algebra. This completes the proof.

$\square$

The above result also shows that the real-valued optimization problem derived from the relaxation is actually a lower bound of the original combinatorial optimization problem. Unlike 2-way partitioning however, it is unclear how to utilize multiple eigenvectors simultaneously to obtain a $k$-way partition. Many heuristics have been proposed in the situation of simple graphs, and they can be applied here as well. Perhaps the most popular one among them is as follows [14]. First form a matrix $X = [\Phi_1 \cdots \Phi_k]$, where $\Phi_i$'s are the eigenvectors of $\Delta$ associated with the $k$ smallest eigenvalues. And then the row vectors of $X$ are regarded as the representations of the graph vertices in $k$-dimensional Euclidian space. Those vectors corresponding to the vertices are generally expected to be well separated, and consequently we can obtain a good partition simply by running $k$-means on them once. [18] has resorted to a semidefinite relaxation model for the $k$-way normalized cut instead of the relatively loose spectral relaxation, and then obtained a more accurate solution. It sounds reasonable to expect that the improved solution will lead to improved clustering. As reported in [18], however, the expected improvement does not occur in practice.

## 7    Transductive inference

We have established algorithms for spectral hypergraph clustering and embedding. Now we consider transductive inference on hypergraphs. Specifically, given a hypergraph $G = (V, E, w)$, the vertices in a subset $S \subset V$ have labels in $L = \{1, -1\}$, our task is to predict the labels of the remaining unlabeled vertices. Basically, we should try to assign the same label to all vertices contained in the same hyperedge. It is actually straightforward to derive a transductive inference approach from a clustering scheme. Let $f : V \mapsto \mathbb{R}$ denote a classification function, which assigns a label sign $f(v)$ to a vertex $v \in V$. Given an objective functional $\Omega(\cdot)$ from some clustering approach, one may choose a classification function by

$$\operatorname*{argmin}_{f \in \mathbb{R}^{|V|}} \{R_{\mathrm{emp}}(f) + \mu\Omega(f)\},$$

where $R_{\mathrm{emp}}(f)$ denotes a chosen empirical loss, such as the least square loss or the hinge loss, and the number $\mu > 0$ the regularization parameter. Since in general normalized cuts are thought to be superior to mincuts, the transductive inference approach that we used in the later experiments is built on the above spectral hypergraph clustering method. Consequently, as shown in [20], with the least square loss function, the classification function is finally given by $f = (I - \xi\Theta)^{-1}y$, where the elements of $y$ denote the initial labels, and $\xi$ is a parameter in $(0, 1)$. For a survey on transductive inference, we refer the readers to [21].

## 8    Experiments

All datasets except a particular version of the 20-newsgroup one are from the UCI Machine Learning Depository. They are usually referred to as the so-called *categorical data*. Specifically, each instance in those datasets is described by one or more attributes. Each attribute takes only a small number of values, each corresponding to a specific category. Attribute values cannot be naturally ordered linearly as numerical values can [9]. In our experiments, we constructed a hypergraph for each dataset, where attribute values were regarded as hyperedges. The weights for all hyperedges were simply set to 1. How to choose suitable weights is definitely an important problem requiring additional exploration however. We also constructed a simple graph for each dataset, and the simple graph spectral clustering based approach [19] was then used as the baseline. Those simple graphs were constructed in the way discussed in the beginning of Section 1, which is essentially to define pairwise relationships among the objects by the adjacency matrices of hypergraphs. The first task we addressed is to embed the animals in the `zoo` dataset into Euclidean space. This dataset contains 100 animals with 17 attributes. The attributes include `hair`, `feathers`, `eggs`, `milk`, `legs`, `tail`, etc. The animals have been manually classified into 7 different categories. We embedded those animals into Euclidean space by using the eigenvectors of the hypergraph Laplacian associated with the smallest eigenvalues (Figure 2). For the animals having the same attributes, we randomly chose one as their representative to put in the figures. It is apparent that those animals are well separated in their Euclidean representations. Moreover, it deserves a further look that `seal` and `dolphin` are significantly

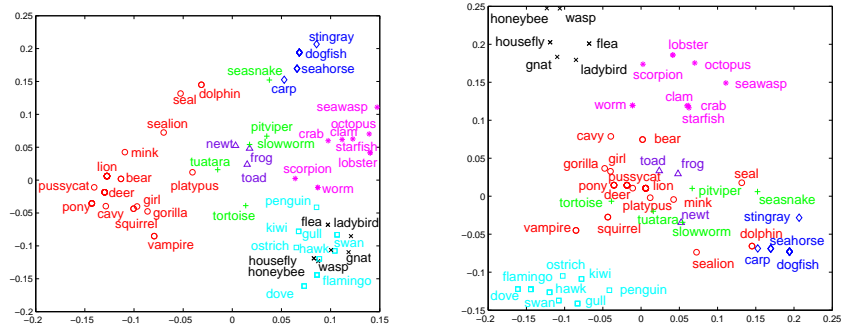

Figure 2: Embedding the `zoo` dataset. Left panel: the eigenvectors with the 2nd and 3rd smallest eigenvalues; right panel: the eigenvectors with the 3rd and 4th smallest eigenvalues. Note that `dolphin` is between class 1 (denoted by ○) containing the animals having milk and living on land, and class 4 (denoted by ◇) containing the animals living in sea.

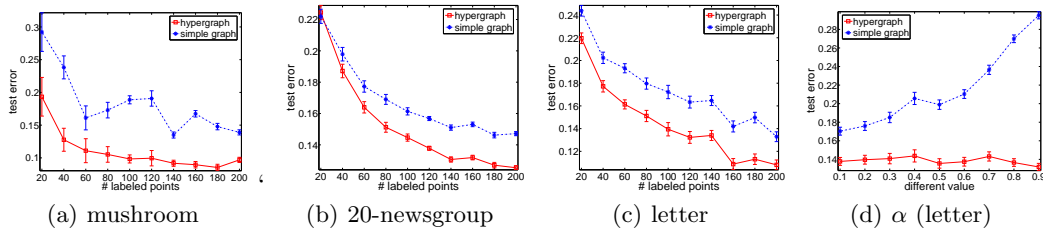

Figure 3: Classification on complex relational data. (a)-(c) Results from both the hypergraph based approach and the simple graph based approach. (d) The influence of the $\alpha$ in letter recognition with 100 labeled instances.

mapped to the positions between class 1 consisting of the animals having milk and living on land, and class 4 consisting of the animals living in sea. A similar observation also holds for `seasnake`. The second task is classification on the `mushroom` dataset that contains 8124 instances described by 22 categorical attributes, such as `shape`, `color`, etc. We remove the 11th attribute that has missing values. Each instance is labeled as `edible` or `poisonous`. They contain 4208 and 3916 instances separately. The third task is text categorization on a modified 20-newsgroup dataset with binary occurrence values for 100 words across 16242 articles (see `http://www.cs.toronto.edu/~roweis`). The articles belong to 4 different topics corresponding to the highest level of the original 20 newsgroups, with the sizes being $4605, 3519, 2657$ and $5461$ respectively. The final task is to guess the letter categories with the `letter` dataset, in which each instance is described by 16 primitive numerical attributes (statistical moments and edge counts). We used a subset containing the instances of the letters from A to E with the sizes being $789, 766, 736, 805$ and $768$ respectively. The experimental results of the above three tasks are shown in Figures 3(a)-3(c). The regularization parameter $\alpha$ is fixed at 0.1. Each testing error is averaged over 20 trials. The results show that the hypergraph based method is consistently better than the baseline. The influence of the $\alpha$ used in the letter recognition task is shown in Figure 3(d). It is interesting that the $\alpha$ influences the baseline much more than the hypergraph based approach.

## 9 Conclusion

We generalized spectral clustering techniques to hypergraphs, and developed algorithms for hypergraph embedding and transductive inference. It is interesting to consider applying the present methodology to a broader range of practical problems. We are particularly interested in the following problems. One is biological network analysis [17]. Biological networks are

mainly modeled as simple graphs so far. It might be more sensible to model them as hypergraphs instead such that complex interactions will be completely taken into account. The other is social network analysis. As recently pointed out by [3], many social transactions are supra-dyadic; they either involve more than two actors or they involve numerous aspects of the setting of interaction. So standard network techniques are not adequate in analyzing these networks. Consequently, they resorted to the concept of a hypergraph, and showed how the concept of network centrality can be adapted to hypergraphs.

## References

[1] S. Agarwal, L. Zelnik-Manor J. Lim, P. Perona, D. Kriegman, and S. Belongie. Beyond pairwise clustering. In *IEEE Conf. on Computer Vision and Pattern Recognition*, 2005.

[2] C. Berge. *Hypergraphs*. North-Holland, Amsterdam, 1989.

[3] P. Bonacich, A.C. Holdren, and M. Johnston. Hyper-edges and multi-dimensional centrality. *Social Networks*, 26(3):189–203, 2004.

[4] P.K. Chan, M.D.F. Schlag, and J. Zien. Spectral k-way ratio cut partitioning and clustering. *IEEE Trans. on Computer Aided Design of Integrated Circuits and Systems*, 13(9):1088–1096, 1994.

[5] F. Chung. *Spectral Graph Theory*. Number 92 in CBMS Regional Conference Series in Mathematics. American Mathematical Society, Providence, RI, 1997.

[6] A. Corduneanu and T. Jaakkola. Distributed information regularization on graphs. In *Advances in Neural Information Processing Systems 17*, Cambridge, MA, 2005. MIT Press.

[7] M. Fiedler. Algebraic connectivity of graphs. *Czechoslovak Mathematical Journal*, 23(98):298–305, 1973.

[8] G. Gallo, G. Longo, and S. Pallottino. Directed hypergraphs and applications. *Discrete Applied Mathematics*, 42(2):177–201, 1993.

[9] D. Gibson, J. Kleinberg, and P. Raghavan. Clustering categorical data: An approach based on dynamical systems. *VLDB Journal*, 8(3-4):222–236, 2000.

[10] M. Gu, H. Zha, C. Ding, X. He, and H. Simon. Spectral relaxation models and structure analysis for k-way graph clustering and bi-clustering. Technical Report CSE-01-007, Department of Computer Science and Engineering, Pennsylvania State University, 2001.

[11] L. Hagen and A.B. Kahng. New spectral methods for ratio cut partitioning and clustering. *IEEE Trans. on Computed-Aided Desgin of Integrated Circuits and Systems*, 11(9):1074–1085, 1992.

[12] D. Klein and C. Manning. Parsing and hypergraphs. In *Proc. 7th Intl. Workshop on Parsing Technologies*, 2001.

[13] M. Meila and J. Shi. A random walks view of spectral segmentation. In *Proc. 8th Intl. Workshop on Artificial Intelligence and Statistics*, 2001.

[14] A.Y. Ng, M.I. Jordan, and Y. Weiss. On spectral clustering: analysis and an algorithm. In *Advances in Neural Information Processing Systems 14*, Cambridge, MA, 2002. MIT Press.

[15] J.S. Oliveira, J.B. Jones-Oliveira, D.A. Dixon, C.G. Bailey, and D.W. Gull. Hyperdigraph–Theoretic analysis of the EGFR signaling network: Initial steps leading to GTP: Ras complex formation. *Journal of Computational Biology*, 11(5):812–842, 2004.

[16] J. Shi and J. Malik. Normalized cuts and image segmentation. *IEEE Tran. on Pattern Analysis and Machine Intelligence*, 22(8):888–905, 2000.

[17] K. Tsuda. Propagating distributions on a hypergraph by dual information regularization. In *Proc. 22th Intl. Conf. on Machine Learning*, 2005.

[18] E.P. Xing and M.I. Jordan. On semidefinite relaxation for normalized k-cut and connections to spectral clustering. Technical Report CSD-03-1265, Division of Computer Science, University of California, Berkeley, 2003.

[19] D. Zhou, O. Bousquet, T.N. Lal, J. Weston, and B. Schölkopf. Learning with local and global consistency. In *Advances in Neural Information Processing Systems 16*, Cambridge, MA, 2004. MIT Press.

[20] D. Zhou, J. Huang, and B. Schölkopf. Learning from labeled and unlabeled data on a directed graph. In *Proc. 22th Intl. Conf. on Machine Learning*, 2005.

[21] X. Zhu. Semi-supervised learning literature survey. Technical Report Computer Sciences 1530, University of Wisconsin - Madison, 2005.
